# Direct Optimization of Margins Improves Generalization in Combined Classifiers

**Llew Mason,Peter Bartlett, Jonathan Baxter**
Department of Systems Engineering
Australian National University, Canberra, ACT 0200, Australia
{lmason, bartlett, jon}@syseng.anu.edu.au

## Abstract

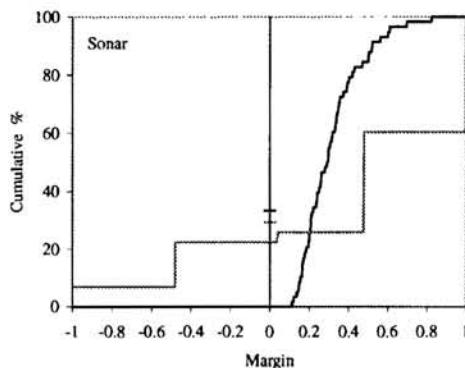

Cumulative training margin distributions for AdaBoost versus our "Direct Optimization Of Margins" (DOOM) algorithm. The dark curve is AdaBoost, the light curve is DOOM. DOOM sacrifices significant training error for improved test error (horizontal marks on margin= 0 line).

## 1 Introduction

Many learning algorithms for pattern classification minimize some cost function of the training data, with the aim of minimizing error (the probability of misclassifying an example). One example of such a cost function is simply the classifier's error on the training data. Recent results have examined alternative cost functions that provide better error estimates in some cases. For example, results in [Bar98] show that the error of a sigmoid network classifier $f(\cdot)$ is no more than the sample average of the cost function $\mathrm{sgn}(\theta - yf(x))$ (which takes value 1 when $yf(x)$ is no more than $\theta$ and 0 otherwise) plus a complexity penalty term that scales as $\|w\|_1/\theta$, where $(x,y) \in X \times \{\pm 1\}$ is a labelled training example, and $\|w\|_1$ is the sum of the magnitudes of the output node weights. The quantity $yf(x)$ is the *margin* of the real-valued function $f$, and reflects the extent to which $f(x)$ agrees with the label $y \in \{\pm 1\}$. By minimizing squared error, neural network learning algorithms implicitly maximize margins, which may explain their good generalization performance.

More recently, Schapire *et al* [SFBL98] have shown a similar result for convex combinations of classifiers, such as those produced by boosting algorithms. They show

that, with high probability over $m$ random examples, every convex combination of classifiers from some finite class $H$ has error satisfying

$$\Pr[yf(x) \le 0] \le \mathbf{E}_S\left[\operatorname{sgn}(\theta - yf(x))\right] + O\left(\frac{1}{\sqrt{m}}\left(\frac{\log m \log|H|}{\theta^2} + \log(1/\delta)\right)^{\frac{1}{2}}\right) \quad (1)$$

for all $\theta > 0$, where $\mathbf{E}_S$ denotes the average over the sample $S$.

One way to think of these results is as a technique for adjusting the effective complexity of the function class by adjusting $\theta$. Large values of $\theta$ correspond to low complexity and small values to high complexity. If the learning algorithm were to optimize the parametrized cost function $\mathbf{E}_S\operatorname{sgn}(\theta - yf(x))$ for large values of $\theta$, it would not be able to make fine distinctions between different functions in the class, and so the effective complexity of the class would be reduced. The second term in the error bounds (the regularization term involving the complexity parameter $\theta$ and the size of the base hypothesis class $H$) would be correspondingly reduced. In both the neural network and boosting settings, the learning algorithms do not directly minimize these cost functions; we use different values of the complexity parameter in the cost functions only in explaining their generalization performance.

In this paper, we address the question: what are suitable cost functions for convex combinations of classifiers? In the next section, we give general conditions on parametrized families of cost functions that ensure that they can be used to give error bounds for convex combinations of classifiers. In the remainder of the paper, we investigate learning algorithms that choose the convex coefficients of a combined classifier by minimizing a suitable family of piecewise linear cost functions using gradient descent. Even when the base hypotheses are chosen by the AdaBoost algorithm, and we only use the new cost functions to adjust the convex coefficients, we obtained an improvement on the test error of AdaBoost in all but one of the UC Irvine data sets we used. Margin distribution plots show that in many cases the algorithm achieves these lower errors by sacrificing training error, in the interests of reducing the new cost function.

## 2 Theory

In this section, we derive an error bound that generalizes the result for convex combinations of classifiers described in the previous section. The result involves a family of *margin cost functions* (functions mapping from the interval $[-1, 1]$ to $\mathbb{R}^+$), indexed by an integer-valued complexity parameter $N$, which measures the resolution at which we examine the margins. The following definition gives conditions on the margin cost functions that relate the complexity $N$ to the amount by which the margin cost function is larger than the function $\operatorname{sgn}(-yf(x))$. The particular form of this definition is not important. In particular, the functions $\Psi_N$ are only used in the analysis in this section, and will not concern us later in the paper.

**Definition 1** *A family $\{C_N : N \in \mathbb{N}\}$ of margin cost functions is $B$-admissible for $B \ge 0$ if for all $N \in \mathbb{N}$ there is an interval $Y \subset \mathbb{R}$ of length no more than $B$ and a function $\Psi_N : [-1, 1] \to Y$ that satisfies*

$$\operatorname{sgn}(-\alpha) \le \mathbf{E}_{Z \sim Q_{N,\alpha}}\left(\Psi_N(Z)\right) \le C_N(\alpha)$$

*for all $\alpha \in [-1, 1]$, where $\mathbf{E}_{Z \sim Q_{N,\alpha}}(\cdot)$ denotes the expectation when $Z$ is chosen randomly as $Z = (1/N)\sum_{i=1}^{N} Z_i$ with $Z_i \in \{-1, 1\}$ and $\Pr(Z_i = 1) = (1 + \alpha)/2$.*

As an example, let $C_N(\alpha) = \operatorname{sgn}(\theta - \alpha) + c$, for $\theta = 1/\sqrt{N}$ and some constant $c$. This is a $B$-admissible family of margin cost functions, for suitably large $B$. (This is

exhibited by the functions $\Psi_N(\alpha) = \text{sgn}(\theta/2 - \alpha) + c/2$; the proof involves Chernoff bounds.) Clearly, for larger values of $N$, the cost functions $C_N$ are closer to the threshold function $\text{sgn}(-\alpha)$. Inequality (1) is implied by the following theorem. In this theorem, $\text{co}(H)$ is the set of convex combinations of functions from $H$. A similar proof gives the same result with $\text{VCdim}(H) \ln m$ replacing $\ln |H|$.

**Theorem 2** *For any B-admissible family $\{C_N : N \in \mathbb{N}\}$ of margin cost functions, any finite hypothesis class $H$ and any distribution $P$ on $X \times \{-1, 1\}$, with probability at least $1 - \delta$ over a random sample $S$ of $m$ labelled examples chosen according to $P$, every $N$ and every $f$ in $\text{co}(H)$ satisfies*

$$\Pr[yf(x) \leq 0] < \mathbf{E}_S[C_N(yf(x))] + \sqrt{\frac{B^2}{2m}(N \ln |H| + \ln(N(N+1)/\delta))}.$$

**Proof** Fix $N$ and $f \in \text{co}(H)$, and suppose that $f = \sum_i \alpha_i h_i$ for $h_i \in H$. Define $\text{co}_N(H) = \left\{ (1/N) \sum_{j=1}^{N} h_j : h_j \in H \right\}$, and notice that $|\text{co}_N(H)| \leq |H|^N$. As in the proof of (1) in [SFBL98], we show using the probabilistic method that there is a function $g$ in $\text{co}_N(H)$ that closely approximates $f$. Let $Q$ be the distribution on $\text{co}_N(H)$ corresponding to the average of $N$ independent draws from $\{h_i\}$ according to the distribution $\{\alpha_i\}$, and let $Q_{N,\alpha}$ be the distribution given in Definition 1. Then for any fixed pair $x, y$, when $g$ is chosen according to $Q$ the distribution of $yg(x)$ is $Q_{N,yf(x)}$. Now, fix the function $\Psi_N$ implied by the $B$-admissibility condition. By the definition of $B$-admissibility,

$$\mathbf{E}_{g \sim Q}\mathbf{E}_P[\Psi_N(yg(x))] = \mathbf{E}_P \mathbf{E}_{Z \sim Q_{N,yf(x)}}[\Psi_N(Z)] \geq \mathbf{E}_P \text{sgn}(-yf(x)) = P[yf(x) \leq 0].$$

Similarly, $\mathbf{E}_S[C_N(yf(x))] \geq \mathbf{E}_{g \sim Q}\mathbf{E}_S[\Psi_N(yg(x))]$. Hence, if $\Pr[yf(x) \leq 0] - \mathbf{E}_S[C_N(yf(x))] \geq \epsilon_N$, then $\mathbf{E}_{g \sim Q}[\mathbf{E}_P[\Psi_N(yg(x))] - \mathbf{E}_S[\Psi_N(yg(x))]] \geq \epsilon_N$. Thus,

$$\Pr[\exists f \in \text{co}(H): \Pr[yf(x) \leq 0] \geq \mathbf{E}_S[C_N(yf(x))] + \epsilon_N]$$
$$\leq \Pr[\exists g \in \text{co}_N(H): \mathbf{E}_P[\Psi_N(yg(x))] \geq \mathbf{E}_S[\Psi_N(yg(x))] + \epsilon_N]$$
$$\leq |H|^N \exp(-2m\epsilon_N^2/B^2),$$

where the last inequality follows from the union bound and Hoeffding's inequality. Setting this probability to $\delta_N = \delta/(N(N+1))$, solving for $\epsilon_N$, and summing over values of $N$ completes the proof, since $\sum_{N \in \mathbb{N}} \delta_N = \delta$. □

For the best bounds, we want $\Psi_N$ to satisfy $\mathbf{E}_{Z \sim Q_{N,\alpha}}[\Psi_N(Z)] \geq \text{sgn}(-\alpha)$, but with the difference $\mathbf{E}_{Z \sim Q_{N,\alpha}}[\Psi_N(Z) - \text{sgn}(-\alpha)]$ as small as possible for $\alpha \in [-1, 1]$. One approach would be to minimize the expectation of this difference, for $\alpha$ chosen uniformly in $[-1, 1]$. However, this yields a non-monotone solution for $C_N(\alpha)$. Figure 1a illustrates an example of a monotone $B$-admissible family; it shows the cost functions $C_N(\alpha) = \mathbf{E}_{Z \sim Q_{N,\alpha}} \Psi_N(Z)$, for $N = 20, 50$ and $200$, where $\Psi_N(\alpha) = \text{sgn}(\sqrt{2 \log N/N} - \alpha) + 1/N$.

## 3   Algorithm

We now consider how to select convex coefficients $w_1, \ldots, w_T$ for a sequence of $\{-1, 1\}$ classifiers $h_1, \ldots, h_T$ so that the combined classifier $f(x) = \sum_{t=1}^{T} w_t h_t(x)$ has small error. In the experiments we used the hypotheses provided by AdaBoost. (The aim was to investigate how useful are the error estimates provided by the cost functions of the previous section.)

If we take Theorem 2 at face value and ignore log terms, the best error bound is obtained if the weights $w_1, \ldots, w_T$ and the complexity $N$ are chosen to minimize

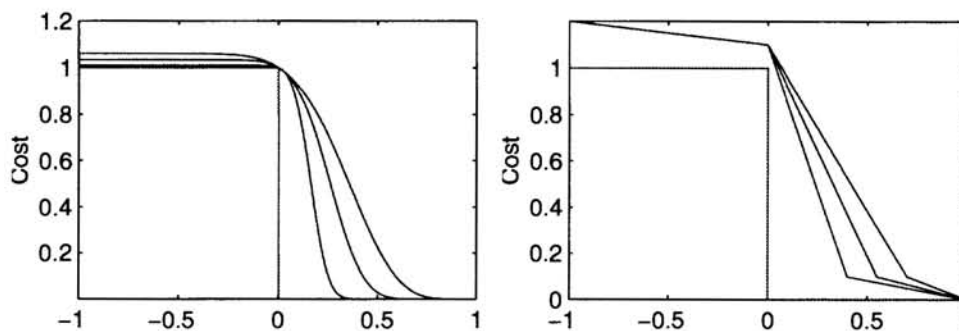

Figure 1: **(a)** The margin cost functions $C_N(\alpha)$, for $N = 20, 50$ and $200$, compared to the function $\mathrm{sgn}(-\alpha)$. Larger values of $N$ correspond to closer approximations to $\mathrm{sgn}(-\alpha)$. **(b)** Piecewise linear upper bounds on the functions $C_N(\alpha)$, and the function $\mathrm{sgn}(-\alpha)$.

$(1/m)\sum_{i=1}^{m} C_N(y_i f(x_i)) + \kappa\sqrt{N/m}$, where $\kappa$ is a constant and $\{C_N\}$ is a family of $B$-admissible cost functions. Although Theorem 2 provides an expression for the constant $\kappa$, in practical problems this will almost certainly be an overestimate and so our penalty for even moderately complex models will be too great. To solve this problem, instead of optimizing the average cost of the margins plus a penalty term over all values of the parameter $\theta$, we estimated the optimal value of $\theta$ using a cross-validation set. That is, for fixed values of $\theta$ in a discrete but fairly dense set we selected weights optimizing the average cost $\frac{1}{m}\sum_{i=1}^{m} C_\theta(y_i f(x_i))$ and then chose the solution with smallest error on an independent cross-validation set.

We considered the use of the cost functions plotted in Figure 1a, but the existence of flat regions caused difficulties for gradient descent approaches. Instead we adopted a piecewise linear family of cost functions $C_\theta$ that are linear in the intervals $[-1, 0]$, $[0, \theta]$, and $[\theta, 1]$, and pass through the points $(-1, 1.2)$, $(0, 0.1)$, $(\theta, 0.1)$, and $(1, 0)$, for $\theta \in (0, 1)$. The numbers were chosen to ensure the $C_\theta$ are upper bounds on the cost functions of Figure 1a (see Figure 1b). Note that $\theta$ plays the role of a complexity parameter, except that in this case smaller values of $\theta$ correspond to higher complexity classes.

Even with the restriction to piecewise linear cost functions, the problem of optimizing $\frac{1}{m}\sum_{i=1}^{m} C_\theta(y_i f(x_i))$ is still hard. Fortunately, the nature of this cost function makes it possible to find successful heuristics (which is why we chose it). The algorithm we have devised to optimize the $C_\theta$ family of cost functions is called Direct Optimization Of Margins (DOOM). (The pseudo-code of the algorithm is given in the full version [MBB98].) DOOM is basically a form of gradient descent, with two complications: it takes account of the fact that the cost function is not differentiable at 0 and $\theta$, and it ensures that the weight vector lies on the unit ball in $l_1$. In order to avoid problems with local minima we actually allow the weight vector to lie within the $l_1$-ball throughout optimization rather than on the $l_1$-ball. If the weight vector reaches the surface of the $l_1$-ball and the update direction points out of the $l_1$-ball, it is projected back to the surface of the $l_1$-ball.

Observe that the gradient of $\frac{1}{m}\sum_{i=1}^{m} C_\theta(y_i f(x_i))$ is a constant function of the weights $w = (w_1, \ldots, w_T)$ provided no example $(x_i, y_i)$ "crosses" one of the discontinuities at 0 or $\theta$ (i.e. provided the margin $y_i f(x_i)$ does not cross 0 or $\theta$). Hence, the central operation of DOOM is to step in the negative gradient direction until an example's margin hits one of the discontinuities (projecting where necessary to ensure the weight vector lies within the $l_1$ ball). At this point the gradient vector becomes multi-valued (generally two-valued but it can be more). Each of the possible gradient directions is then tested by taking a small step in that direction (a

random subset of the gradient directions is chosen if there are too many of them). If none of the directions lead to a decrease in the cost, the examples whose margins lie on discontinuities of the cost function are added to a constraint set $E$. In subsequent iterations the same stepping procedure above is followed except that the direction step is modified to ensure that the examples in $E$ do not move (i.e. they remain on the discontinuity points of $C_\theta$). That is, the weight vector moves within the subspace defined by the examples in $E$. If no progress is made in any iteration, the constraint set $E$ is reset to zero. If still no progress is made the procedure terminates.

# 4   Experiments

We used the following two-class problems from the UC Irvine database [CBM98] : Cleveland Heart Disease, Credit Application, German, Glass, Ionosphere, King Rook vs King Pawn, Pima Indians Diabetes, Sonar, Tic-Tac-Toe, and Wisconsin Breast Cancer. For the sake of simplicity we did not consider multi-class problems. Each data set was randomly separated into train, test and validation sets, with the test and validation sets being equal in size. This was repeated 10 times independently and the results were averaged.

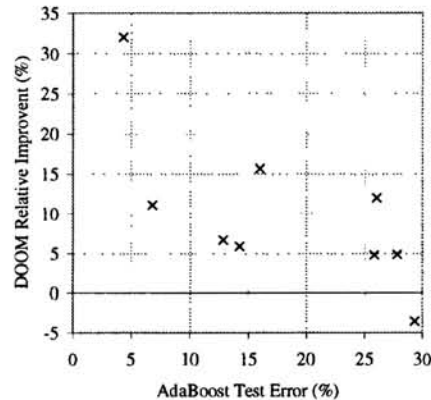

Figure 2: Relative improvement of DOOM over AdaBoost for all examined datasets.

Each experiment consisted of the following steps. First, AdaBoost was run on the training data to produce a sequence of base classifiers and their corresponding weights. In all of the experiments the base classifiers were axis-orthogonal hyperplanes (also known as decision stumps); this choice ensured that the complexity of the class of base classifiers was constant. Boosting was halted when adding a new classifier failed to decrease the error on the validation set. DOOM was then run on the classifiers produced by AdaBoost for a large range of $\theta$ values and 1000 random initial weight vectors for each value of $\theta$. The weight vector (and $\theta$ value) with minimum misclassification on the validation set was chosen as the final solution.

In some cases the training sets were reduced in size to make overfitting more likely, so that complexity regularization with DOOM could have an effect. (The details are given in the full version [MBB98].) In three of the datasets (Credit Application, Wisconsin Breast Cancer and Pima Indians Diabetes), AdaBoost gained no advantage from using more than a single classifier. In these datasets, the number of classifiers was chosen so that the validation error was reasonably stable.

A comparison between the test errors generated by AdaBoost and DOOM is shown in Figure 2. In only one data set did DOOM produce a classifier which performed worse than AdaBoost in terms of test error; for most data sets DOOM's test error was a significant improvement over AdaBoost's.

Figure 3 shows cumulative training margin distribution graphs for four of the datasets for both AdaBoost and DOOM (with optimal $\theta$ chosen by cross-validation). For a given margin the value on the curve corresponds to the proportion of training examples with margin no more than this value. The test errors for both algorithms are also shown for comparison, as short horizontal lines on the vertical axis.

The margin distributions show that the value of the minimum training margin has no real impact on generalization performance. (See also [Bre97] and [GS98].) As

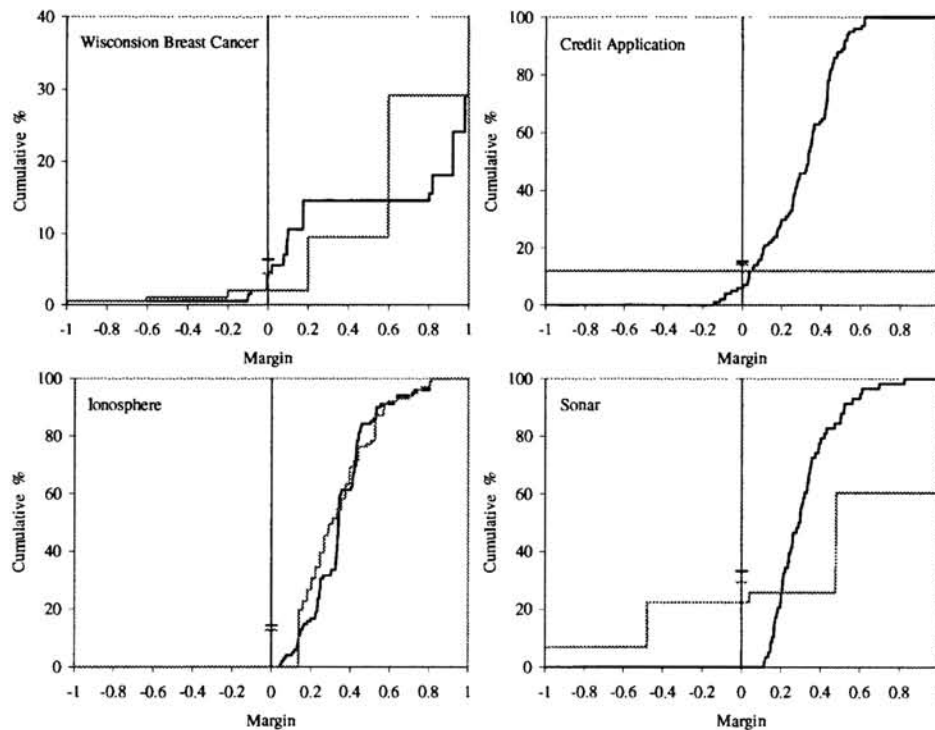

Figure 3: Cumulative training margin distributions for four datasets. The dark curve is AdaBoost, the light curve is DOOM with $\theta$ selected by cross-validation. The test errors for both algorithms are marked on the vertical axis at margin 0.

can be seen in Figure 3 (Credit Application and Sonar data sets), the generalization performance of the combined classifier produced by DOOM can be as good as or better than that of the classifier produced by AdaBoost, despite having dramatically worse minimum training margin. Conversely, Figure 3 (Ionosphere data set) shows that improved generalization performance can be associated with an improved minimum margin.

The margin distributions also show that there is a balance to be found between training error and complexity (as measured by $\theta$). DOOM is willing to sacrifice training error in order to reduce complexity and thereby obtain a better margin distribution. For instance, in Figure 3 (Sonar data set), DOOM's training error is over 20% while AdaBoost's is 0%, but DOOM's test error is 5% less than that of AdaBoost's. The reason for this success can be seen in Figure 4, which illustrates the changes in the cost function, training error, and test error as a function of $\theta$. The optimal complexity for this data set is low (corresponding to a large optimal $\theta$). In this case, a reduction in complexity is more important to generalization error than a reduction in training error.

## 5 Conclusion

In this paper we have addressed the question: what are suitable cost functions for convex combinations of base hypotheses? For general families of cost functions that are functions of the *margin* of a sample, we proved (Theorem 2) that the error of a convex combination is no more than the sample average of the cost function plus a regularization term involving the complexity of the cost function and the size of the base hypothesis class.

We constructed a piecewise linear family of cost functions satisfying the conditions of Theorem 2 and presented a heuristic algorithm (DOOM) for optimizing the sample

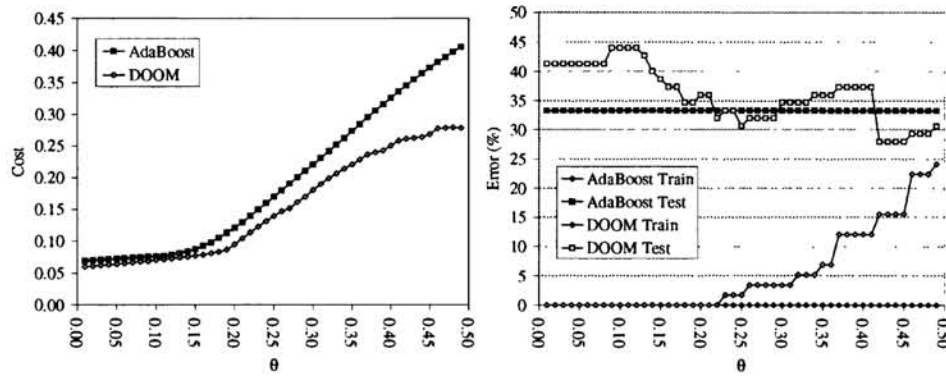

Figure 4: Sonar data set. **Left:** Plot of cost $(\frac{1}{m}\sum_{i=1}^{m} C_\theta(y_i f(x_i)))$ against $\theta$ for AdaBoost and DOOM. **Right:** Plot of training and test error against $\theta$.

average of the cost.

We ran experiments on several of the datasets in the UC Irvine database, in which AdaBoost was used to generate a set of base classifiers and then DOOM was used to find the optimal convex combination of those classifiers. In all but one case the convex combination generated by DOOM had lower test error than AdaBoost's combination. Margin distribution plots show that in many cases DOOM achieves these lower test errors by sacrificing training error, in the interests of reducing the new cost function. The margin plots also show that the size of the minimum margin is not relevant to generalization performance.

## Acknowledgments

Thanks to Yoav Freund, Wee Sun Lee and Rob Schapire for helpful comments and suggestions. This research was supported in part by a grant from the Australian Research Council. Jonathan Baxter was supported by an Australian Research Council Fellowship and Llew Mason was supported by an Australian Postgraduate Award.

## References

[Bar98]    P. L. Bartlett. The sample complexity of pattern classification with neural networks: the size of the weights is more important than the size of the network. *IEEE Transactions on Information Theory*, 44(2):525–536, 1998.

[Bre97]    L. Breiman. Prediction games and arcing algorithms. Technical Report 504, Department of Statistics, University of California, Berkeley, 1997.

[CBM98]  E. Keogh C. Blake and C.J. Merz. UCI repository of machine learning databases, 1998. http://www.ics.uci.edu/~mlearn/MLRepository.html.

[GS98]    A. Grove and D. Schuurmans. Boosting in the limit: Maximizing the margin of learned ensembles. In *Proceedings of the Fifteenth National Conference on Artificial Intelligence*, pages 692–699, 1998.

[MBB98]  L. Mason, P. L. Bartlett, and J. Baxter. Improved generalization through explicit optimization of margins. Technical report, Department of Systems Engineering, Australian National University, 1998. (Available from http://syseng.anu.edu.au/lsg).

[SFBL98]  R. E. Schapire, Y. Freund, P. L. Bartlett, and W. S. Lee. Boosting the margin: a new explanation for the effectiveness of voting methods. *Annals of Statistics*, (to appear), 1998.